# Semi-Supervised Learning with Adversarially Missing Label Information

**Umar Syed**      **Ben Taskar**
Department of Computer and Information Science
University of Pennsylvania
Philadelphia, PA 19104
{usyed,taskar}@cis.upenn.edu

## Abstract

We address the problem of semi-supervised learning in an adversarial setting. Instead of assuming that labels are missing at random, we analyze a less favorable scenario where the label information can be missing partially and arbitrarily, which is motivated by several practical examples. We present nearly matching upper and lower generalization bounds for learning in this setting under reasonable assumptions about available label information. Motivated by the analysis, we formulate a convex optimization problem for parameter estimation, derive an efficient algorithm, and analyze its convergence. We provide experimental results on several standard data sets showing the robustness of our algorithm to the pattern of missing label information, outperforming several strong baselines.

## 1 Introduction

Semi-supervised learning algorithms use both labeled and unlabeled examples. Most theoretical analyses of semi-supervised learning assume that $m + n$ labeled examples are drawn i.i.d. from a distribution, and then a subset of size $n$ is chosen uniformly at random and their labels are erased [1]. This *missing-at-random* assumption is best suited for a situation where the labels are acquired by annotating a random subset of all available data. But in many applications of semi-supervised learning, the partially-labeled data is "naturally occurring", and the learning algorithm has no control over which examples were labeled.

For example, pictures on popular websites like Facebook and Flikr are tagged by users at their discretion, and it is difficult to know how users decide which pictures to tag. A similar problem occurs when data is submitted to an online labor marketplace, such as Amazon Mechanical Turk, to be manually labeled. The workers who label the data are often poorly motivated, and may deliberately skip examples that are difficult to correctly label. In such a setting, a learning algorithm should not assume that the examples were labeled at random.

Additionally, in many semi-supervised learning settings, the partial label information is not provided on a per-example basis. For example, in *multiple instance learning* [2], examples are presented to a learning algorithm in sets, with either zero or one positive examples per set. In *graph-based regularization* [3], a learning algorithm is given information about which examples are likely to have the same label, but not necessarily the identity of that label. Recently, there has been much interest in algorithms that learn from *labeled features* [4]; in this setting, the learning algorithm is given information about the expected value of several features with respect to the true distribution on labeled examples.

To summarize, in a typical semi-supervised learning problem, label information is often missing in an arbitrary fashion, and even when present, does not always have a simple form, like one label per example. *Our goal in this paper is to develop and analyze a learning algorithm that is explicitly*

*designed for these types of problems.* We derive our learning algorithm within a framework that is expressive enough to permit a very general notion of label information, allowing us to make minimal assumptions about which examples in a data set have been labeled, how they have been labeled, and why. We present both theoretical upper and lower bounds for learning in this framework, and motivated by these bounds, derive a simple yet provably optimal learning algorithm. We also provide experimental results on several standard data sets, which show that our algorithm is effective and robust when the label information has been provided by "lazy" or "unhelpful" labelers.

**Related Work:** Our learning framework is related to the *malicious label noise* setting, in which the labeler is allowed to mislabel a small fraction of the training set (this is a special case of the even more challenging *malicious noise* setting [5], where an adverary can inject a small number of arbitrary examples into the training set). Learning with this type of label noise is known to be quite difficult, and positive results often make quite restrictive assumptions about the underlying data distribution [6, 7]. By contrast, our results apply far more generally, at the expense of assuming a more benign (but possibly more realistic) model of label noise, where the labeler can adversarially *erase* labels, but not change them. In other words, we assume that the labeler equivocates, but does not lie. The difference in these assumptions shows up quite clearly in our analysis: As we point out in Section 3, our bounds become vacuous if the labeler is allowed to mislabel data.

In Section 2 we describe how our framework encodes label information in a *label regularization* function, which closely resembles the idea of a *compatibility* function introduced by Balcan & Blum [8]. However, they did not analyze a setting where this function is selected adversarially.

## 2 Learning Framework

Let $\mathcal{X}$ be the set of all possible examples, and $\mathcal{Y}$ the set of all possible labels, where $|\mathcal{Y}| = k$. Let $\mathcal{D}$ be an unknown distribution on $\mathcal{X} \times \mathcal{Y}$. We write $\mathbf{x}$ and $\mathbf{y}$ as abbreviations for $(x_1, \dots, x_m) \in \mathcal{X}^m$ and $(y_1, \dots, y_m) \in \mathcal{Y}^m$, respectively. We write $(\mathbf{x}, \mathbf{y}) \sim \mathcal{D}^m$ to denote that each $(x_i, y_i)$ is drawn i.i.d. from the distribution $\mathcal{D}$ on $\mathcal{X} \times \mathcal{Y}$, and $\mathbf{x} \sim \mathcal{D}^m$ to denote that each $x_i$ is drawn i.i.d. from the marginal distribution of $\mathcal{D}$ on $\mathcal{X}$.

Let $(\hat{\mathbf{x}}, \hat{\mathbf{y}}) \sim \mathcal{D}^m$ be the $m$ labeled training examples. In supervised learning, one assumes access to the entire training set $(\hat{\mathbf{x}}, \hat{\mathbf{y}})$. In semi-supervised learning, one assumes access to only some of the labels $\hat{\mathbf{y}}$, and in most theoretical analyses, the missing components of $\hat{\mathbf{y}}$ are assumed to have been selected uniformly at random.

We make a much weaker assumption about what label information is available. We assume that, after the labeled training set $(\hat{\mathbf{x}}, \hat{\mathbf{y}})$ has been drawn, the learning algorithm is only given access to the examples $\hat{\mathbf{x}}$ and to a *label regularization function* $R$. The function $R$ encodes some information about the labels $\hat{\mathbf{y}}$ of $\hat{\mathbf{x}}$, and is selected by a potentially adversarial *labeler* from a family $\mathcal{R}(\hat{\mathbf{x}}, \hat{\mathbf{y}})$. A label regularization function $R$ maps each possible *soft labeling* $\mathbf{q}$ of the training examples $\hat{\mathbf{x}}$ to a real number $R(\mathbf{q})$ (a soft labeling is natual generalization of a labeling that we will define formally in a moment). Except for knowing that $R$ belongs to $\mathcal{R}(\hat{\mathbf{x}}, \hat{\mathbf{y}})$, the learner can make *no* assumptions about how the labeler selects $R$. We give examples of label regularization functions in Section 2.1.

Let $\Delta$ denote the set of distributions on $\mathcal{Y}$. A soft labeling $\mathbf{q} \in \Delta^m$ of the training examples $\hat{\mathbf{x}}$ is a doubly-indexed vector, where $q(i, y)$ is interpreted as the probability that example $\hat{x}_i$ has label $y \in \mathcal{Y}$. The correct soft labeling has $q(i, y) = \mathbf{1}\{y = \hat{y}_i\}$, where the indicator function $\mathbf{1}\{\cdot\}$ is 1 when its argument is true and 0 otherwise; we overload notation and write $\hat{\mathbf{y}}$ to denote the correct soft labeling.

Although the labeler is possibly adversarial, the family $\mathcal{R}(\hat{\mathbf{x}}, \hat{\mathbf{y}})$ of label regularization functions restricts the choices the labeler can make. We are interested in designing learning algorithms that work well when each $R \in \mathcal{R}(\hat{\mathbf{x}}, \hat{\mathbf{y}})$ assigns a low value to the correct labeling $\hat{\mathbf{y}}$. In the examples we describe in Section 2.1, the correct labeling $\hat{\mathbf{y}}$ will be near the minimum of $R$, *but there will be many other minima and near-minima as well*. This is the sense in which label information is "missing" — it is difficult for any learning algorithm to distinguish among these minima.

We emphasize that, while our algorithms work best when $\hat{\mathbf{y}}$ is close to the minimum of each $R \in \mathcal{R}(\hat{\mathbf{x}}, \hat{\mathbf{y}})$, nothing in our framework *requires* this to be true; in Section 3 we will see that our learning bounds degrade gracefully as this condition is violated.

We are interested in learning a parameterized model that predicts a label $y$ given an example $x$. Let $L(\boldsymbol{\theta}, x, y)$ be the *loss* of parameter $\boldsymbol{\theta} \in \mathbb{R}^d$ with respect to labeled example $(x, y)$. While some of the development in this paper will apply to generic loss functions, but two loss functions that will particularly interest us are the negative log-likelihood of a log-linear model

$$L_{\text{like}}(\boldsymbol{\theta}, x, y) = -\log p_{\boldsymbol{\theta}}(y|x) = -\log \frac{\exp(\boldsymbol{\theta}^T \boldsymbol{\phi}(x, y))}{\sum_{y'} \exp(\boldsymbol{\theta}^T \boldsymbol{\phi}(x, y'))}$$

where $\boldsymbol{\phi}(x, y) \in \mathbb{R}^d$ is the feature function, and the 0-1 loss of a linear classifier

$$L_{0,1}(\boldsymbol{\theta}, x, y) = \mathbf{1}\{\arg\max_{y' \in \mathcal{Y}} \boldsymbol{\theta}^T \boldsymbol{\phi}(x, y') \neq y\}.$$

Given training examples $\hat{\mathbf{x}}$, label regularization function $R$, and loss function $L$, the goal of a learning algorithm is to find a parameter $\boldsymbol{\theta}$ that minimizes the expected loss $E_{\mathcal{D}}[L(\boldsymbol{\theta}, x, y)]$, where $E_{\mathcal{D}}[\cdot]$ denotes expectation with respect to $(x, y) \sim \mathcal{D}$.

Let $E_{\hat{\mathbf{x}}, \mathbf{q}}[f(x, y)]$ denote the expected value of $f(x, y)$ when example $x$ is chosen uniformly at random from the training examples $\hat{\mathbf{x}}$ and — supposing that this is example $\hat{x}_i$ — label $y$ is chosen from the distribution $\mathbf{q}(i, \cdot)$. Accordingly, $E_{\hat{\mathbf{x}}, \hat{\mathbf{y}}}[f(x, y)]$ denotes the expected value of $f(x, y)$ when labeled example $(x, y)$ is chosen uniformly at random from the labeled training examples $(\hat{\mathbf{x}}, \hat{\mathbf{y}})$.

## 2.1 Examples of Label Regularization Functions

To make the concept of a label regularization function more clear, we describe several well-known learning settings in which the information provided to the learning algorithm is less than the fully labeled training set. We show that, for each these settings, there is a natural definition of $\mathcal{R}$ that captures the information that is provided to the learning algorithm, and thus each of these settings can be seen as special cases of our framework.

Before proceeding with the partially labeled cases, we explain how *supervised learning* can be expressed in our framework. In the supervised learning setting, the label of every example in the training set is revealed to the learner. In this setting, the label regularization function family $\mathcal{R}(\hat{\mathbf{x}}, \hat{\mathbf{y}})$ contains a single function $R_{\hat{\mathbf{y}}}$ such that $R_{\hat{\mathbf{y}}}(\mathbf{q}) = 0$ if $\mathbf{q} = \hat{\mathbf{y}}$, and $R_{\hat{\mathbf{y}}}(\mathbf{q}) = \infty$ otherwise.

In the *semi-supervised learning* setting, the labels of only some of the training examples are revealed. In this case, there is a function $R_I \in \mathcal{R}(\hat{\mathbf{x}}, \hat{\mathbf{y}})$ for each $I \subseteq [m]$ such that $R_I(\mathbf{q}) = 0$ if $q(i, y) = \mathbf{1}\{y = \hat{y}_i\}$ for all $i \in I$ and $y \in \mathcal{Y}$, and $R_I(\mathbf{q}) = \infty$ otherwise. In other words, $R_I(\mathbf{q})$ is zero if and only if the soft labeling $\mathbf{q}$ agrees with $\hat{\mathbf{y}}$ on the examples in $I$. This implies that $R_I(\mathbf{q})$ is independent of how $\mathbf{q}$ labels examples not in $I$ — these are the examples whose labels are missing.

In the *ambiguous learning* setting [9, 10], which is a generalization of semi-supervised learning, the labeler reveals a label set $\hat{Y}_i \subseteq \mathcal{Y}$ for each training example $\hat{x}_i$ such that $\hat{y}_i \in \hat{Y}_i$. That is, for each training example, the learning algorithm is given a set of possibile labels the example can have (semi-supervised learning is the special case where each label set has size 1 or $k$). Letting $\hat{Y} = (\hat{Y}_1, \ldots, \hat{Y}_m)$ be all the label sets revealed to the learner, there is a function $R_{\hat{Y}} \in \mathcal{R}(\hat{\mathbf{x}}, \hat{\mathbf{y}})$ for each possible $\hat{Y}$ such that $R_{\hat{Y}}(\mathbf{q}) = 0$ if $\text{supp}(\mathbf{q}_i) \subseteq \hat{Y}_i$ for all $i \in [m]$ and $R_{\mathbf{Y}}(\mathbf{q}) = \infty$ otherwise. Here $\mathbf{q}_i \triangleq \mathbf{q}(i, \cdot)$ and $\text{supp}(\mathbf{q}_i)$ is the support of label distribution $\mathbf{q}_i$. In other words, $R_{\hat{Y}}(\mathbf{q})$ is zero if and only if the soft labeling $\mathbf{q}$ is supported on the sets $\hat{Y}_1, \ldots, \hat{Y}_m$.

The label regularization functions described above essentially give only local information; they specify, for each example in the training set, which labels are possible for that example. In some cases, we may want to allow the labeler to provide more global information about the correct labeling.

One example of providing global information is *Laplacian regularization*, a kind of graph-based regularization [3] that encodes information about which examples are likely to have the same labels. For any soft labeling $\mathbf{q}$, let $\mathbf{q}[y]$ be the $m$-length vector whose $i$th component is $q(i, y)$. The Laplacian regularizer is defined to be $R_{\mathbf{L}}(\mathbf{q}) = \sum_{y \in \mathcal{Y}} \mathbf{q}[y]^T \mathbf{L}(\hat{\mathbf{x}}) \mathbf{q}[y]$, where $\mathbf{L}(\hat{\mathbf{x}})$ is an $m \times m$ positive semi-definite matrix defined so that $R_{\mathbf{L}}(\mathbf{q})$ is large whenever examples in $\hat{\mathbf{x}}$ that are believed to have the same label are assigned different label distributions by $\mathbf{q}$.

Another possibility is *posterior regularization*. Define a feature function $\mathbf{f}(x, y) \in \mathbb{R}^\ell$; these features may or may not be related to the model features $\boldsymbol{\phi}$ defined in Section 2. As noted by several authors

[4, 11, 12], it is often convenient for a labeler to provide information about the expected value of $\mathbf{f}(x, y)$ with respect to the true distribution. A typical posterior regularizer of this type will have the form $R_{\mathbf{f},\mathbf{b}}(\mathbf{q}) = \|E_{\hat{\mathbf{x}},\mathbf{q}}[\mathbf{f}(x, y)] - \mathbf{b}\|_2^2$, where the vector $\mathbf{b} \in \mathbb{R}^\ell$ is the labeler's estimate of the expected value of $\mathbf{f}$. This term penalizes soft labelings $\mathbf{q}$ which cause the expected value of $\mathbf{f}$ on the training set to deviate from $\mathbf{b}$.

Label regularization functions can also be added together. So, for instance, ambiguous learning can be combined with a Laplacian, and in this case the learner is given a label regularization function of the form $R_{\hat{Y}}(\mathbf{q}) + R_{\mathbf{L}}(\mathbf{q})$. We will experiment with these kinds of combined regularization functions in Section 5.

Note that, in all the examples described above, while the correct labeling $\hat{\mathbf{y}}$ is at or close to the minimum of each function $R \in \mathcal{R}(\hat{\mathbf{x}}, \hat{\mathbf{y}})$, there may be many labelings meeting this condition. Again, this is the sense in which label information is "missing".

It is also important to note that we have only specified *what* information the labeler can reveal to the learner (some function from the set $\mathcal{R}(\hat{\mathbf{x}}, \hat{\mathbf{y}})$), but we do not specify *how* that information is chosen by the labeler (which function $R \in \mathcal{R}(\hat{\mathbf{x}}, \hat{\mathbf{y}})$?). This will have a significant impact on our analysis of this framework. To see why, consider the example of semi-supervised learning. Using the notation defined above, most analyses of semi-supervised learning assume that $R_I$ is chosen be selecting a random subset $I$ of the training examples [13, 14]. By constrast, we make no assumptions about how $R_I$ is chosen, because we are interested in settings where such assumptions are not realistic.

## 3   Upper and Lower Bounds

In this section, we state upper and lower bounds for learning in our framework. But first, we provide a definition of the well-known concept of uniform convergence.

**Definition 1** (Uniform Convergence). *Loss function L has $\epsilon$-uniform convergence if with probability $1 - \delta$*

$$\sup_{\boldsymbol{\theta} \in \Theta} \left| E_{\mathcal{D}}[L(\boldsymbol{\theta}, x, y)] - E_{\hat{\mathbf{x}},\hat{\mathbf{y}}}[L(\boldsymbol{\theta}, x, y)] \right| \leq \epsilon(\delta, m)$$

*where $(\hat{\mathbf{x}}, \hat{\mathbf{y}}) \sim \mathcal{D}^m$ and $\epsilon(\cdot, \cdot)$ is an expression bounding the rate of convergence.*

For example, if $\|\boldsymbol{\phi}(x, y)\| \leq c$ for all $(x, y) \in \mathcal{X} \times \mathcal{Y}$ and $\Theta = \{\boldsymbol{\theta} : \|\boldsymbol{\theta}\| \leq 1\} \subseteq \mathbb{R}^d$, then the loss function $L_{\text{like}}$ has $\epsilon$-uniform convergence with $\epsilon(\delta, m) = O\left(c\sqrt{\frac{d \log m + \log(1/\delta)}{m}}\right)$, which follows from standard results about Rademacher complexity and covering numbers. Other commonly used loss functions, such as hinge loss and 0-1 loss, also have $\epsilon$-uniform convergence under similar boundedness assumptions on $\boldsymbol{\phi}$ and $\Theta$.

We are now ready to state an upper bound for learning in our framework. The proof is contained in the supplement.

**Theorem 1.** *Suppose loss function L has $\epsilon$-uniform convergence. If $(\hat{\mathbf{x}}, \hat{\mathbf{y}}) \sim \mathcal{D}^m$ then with probability at least $1 - \delta$ for all parameters $\boldsymbol{\theta} \in \Theta$ and label regularization functions $R \in \mathcal{R}(\hat{\mathbf{x}}, \hat{\mathbf{y}})$*

$$E_{\mathcal{D}}[L(\boldsymbol{\theta}, x, y)] \leq \max_{\mathbf{q} \in \Delta^m} (E_{\hat{\mathbf{x}},\mathbf{q}}[L(\boldsymbol{\theta}, x, y)] - R(\mathbf{q})) + R(\hat{\mathbf{y}}) + \epsilon(\delta, m).$$

Theorem 2 below states a lower bound that nearly matches the upper bound in Theorem 1, in certain cases. As we will see, the existence of a matching lower bound depends strongly on the structure of the label regularization function family $\mathcal{R}$. Note that, given a labeled training set $(\mathbf{x}, \mathbf{y})$, the set $\mathcal{R}(\mathbf{x}, \mathbf{y})$ essentially constrains what information the labeler can reveal to the learning algorithm, thereby encoding our assumptions about how the labeler will behave. We make three such assumptions, described below. For the remainder of this section, we let the set of all possible examples $\mathcal{X} = \{\tilde{x}_1, \ldots, \tilde{x}_N\}$ be finite.

Recall that all the label regularization functions described in Section 2.1 use the value $\infty$ to indicate which labelings of the training set are impossible. Our first assumption is that, for each $R \in \mathcal{R}(\mathbf{x}, \mathbf{y})$, the set of possible labelings under $R$ is *separable* over examples.

**Assumption 1** (∞-Separability). *For all labeled training sets $(\mathbf{x}, \mathbf{y})$ and $R \in \mathcal{R}(\mathbf{x}, \mathbf{y})$ there exists a collection of label sets $\{Y_{\tilde{x}} : \tilde{x} \in \mathcal{X}\}$ and real-valued function $F$ such that $R(\mathbf{q}) = \sum_{i=1}^{m} \chi\{\mathrm{supp}(\mathbf{q}_i) \subseteq Y_{x_i}\} + F(\mathbf{q})$, where the characteristic function $\chi\{\cdot\}$ is 0 when its argument is true and $\infty$ otherwise, and $F(\mathbf{q}) < \infty$ for all $\mathbf{q} \in \Delta^m$.*

It is easy to verify that all the examples of label regularization function families given in Section 2.1 satisfy Assumption 1. Also note that Assumption 1 allows the finite part of $R$ (denoted by $F$) to depend on the entire soft labeling $\mathbf{q}$ in a basically arbitrarily manner.

Before describing our second assumption, we need a few additional definitions. We write $h$ to denote a *labeling function* that maps examples $\mathcal{X}$ to labels $\mathcal{Y}$. Also, for any labeling function $h$ and unlabeled training set $\mathbf{x} \in \mathcal{X}^m$, we let $\mathbf{h}(\mathbf{x}) \in \mathcal{Y}^m$ denote the vector of labels whose $i$th component is $h(x_i)$. Let $\mathbf{p}_\mathbf{x}$ be an $N$-length vector that represents unlabeled training set $\mathbf{x}$ as a distribution on $\mathcal{X}$, whose $i$th component is $\mathbf{p}_\mathbf{x}(i) \triangleq \frac{|\{j : x_j = \tilde{x}_i\}|}{m}$.

Our second assumption is the labeler's behavior is *stable*: If training sets $(\mathbf{x}, \mathbf{y})$ and $(\mathbf{x}', \mathbf{y}')$ are "close" (by which we mean that they are consistently labeled and $\|\mathbf{p}_\mathbf{x} - \mathbf{p}_{\mathbf{x}'}\|_\infty$ is small) then the label regularization functions available to the labeler for each training set are the "same", in the sense that the sets of possible labelings under each of them are identical.

**Assumption 2** ($\gamma$-Stability). *For any labeling function $h^*$ and unlabeled training sets $\mathbf{x}, \mathbf{x}'$ such that $\|\mathbf{p}_\mathbf{x} - \mathbf{p}_{\mathbf{x}'}\|_\infty \leq \gamma$ the following holds: For all $R \in \mathcal{R}(\mathbf{x}, \mathbf{h}^*(\mathbf{x}))$ there exists $R' \in \mathcal{R}(\mathbf{x}', \mathbf{h}^*(\mathbf{x}'))$ such that $R(\mathbf{h}(\mathbf{x})) < \infty$ if and only if $R'(\mathbf{h}(\mathbf{x}')) < \infty$, for all labeling functions $\mathbf{h}$.*

Our final assumption, which we call *reciprocity*, states there is no way to deduce which of the possible labelings under $R$ is the correct one only by examining $R$.

**Assumption 3** (Reciprocity). *For all labeled training sets $(\mathbf{x}, \mathbf{y})$ and $R \in \mathcal{R}(\mathbf{x}, \mathbf{y})$, if $R(\mathbf{y}') < \infty$ then $R \in \mathcal{R}(\mathbf{x}, \mathbf{y}')$.*

Of all our assumptions, reciprocity seems to be the most unnatural and unmotivated. We argue it is necessary for two reasons: Firstly, all the examples of label regularization function families given in Section 2.1 satisfy this assumption, and secondly, in Theorem 3 we show that lifting the reciprocity assumption makes the upper bound in Theorem 1 very loose.

We are nearly ready to state our lower bound. Let $A$ be a (possibly randomized) learning algorithm that takes a set of unlabeled training examples $\hat{\mathbf{x}}$ and a label regularization function $R$ as input, and outputs an estimated parameter $\hat{\boldsymbol{\theta}}$. Also, if under distribution $\mathcal{D}$ each example $x \in \mathcal{X}$ is associated with exactly one label $h^*(x) \in \mathcal{Y}$, then we write $\mathcal{D} = \mathcal{D}_\mathcal{X} \cdot h^*$, where the *data distribution* $\mathcal{D}_\mathcal{X}$ is the marginal distribution of $\mathcal{D}$ on $\mathcal{X}$. Theorem 2 proves the existence of a true labeling function $h^*$ such that a nearly tight lower bound holds for all learning algorithms $A$ *and* all data distributions $\mathcal{D}_\mathcal{X}$ whenever the training set is drawn from $\mathcal{D}_\mathcal{X} \cdot h^*$. The fact that our lower bound holds for all data distributions significantly complicates the analysis, but this generality is important: since $\mathcal{D}_\mathcal{X}$ is typically easy to estimate, it is possible that the learning algorithm $A$ has been tuned for $\mathcal{D}_\mathcal{X}$. The proof of Theorem 2 is contained in the supplement.

**Theorem 2.** *Suppose Assumptions 1, 2 and 3 hold for label regularization function family $\mathcal{R}$, the loss function $L$ is 0-1 loss, and the set of all possible examples $\mathcal{X}$ is finite. For all learning algorithms $A$ and data distributions $\mathcal{D}_\mathcal{X}$ there exists a labeling function $h^*$ such that if $(\hat{\mathbf{x}}, \hat{\mathbf{y}}) \sim \mathcal{D}^m$ (where $\mathcal{D} = \mathcal{D}_\mathcal{X} \cdot h^*$) and $m \geq O(\frac{1}{\gamma^2} \log \frac{|\mathcal{X}|}{\delta})$ then with probability at least $\frac{1}{4} - 2\delta$*

$$E_\mathcal{D}[L(\hat{\boldsymbol{\theta}}, x, y)] \geq \frac{1}{4} \max_{\mathbf{q} \in \Delta^m} \left( E_{\hat{\mathbf{x}}, \mathbf{q}}[L(\hat{\boldsymbol{\theta}}, x, y)] - R(\mathbf{q}) \right) + \min_{\mathbf{q} \in \Delta^m} R(\mathbf{q}) - \epsilon(\delta, m)$$

*for some $R \in \mathcal{R}(\hat{\mathbf{x}}, \hat{\mathbf{y}})$, where $\hat{\boldsymbol{\theta}}$ is the parameter output by $A$, and $\gamma$ is the constant from Assumption 2.*

Obviously, Assumptions 1, 2 and 3 restrict the kinds of label regularization function families to which Theorem 2 can be applied. However, some restriction is necessary in order to prove a meaningful lower bound, as Theorem 3 below shows. This theorem states that if Assumption 3 does not hold, then it may happen that each family $\mathcal{R}(\mathbf{x}, \mathbf{y})$ has a structure which a clever (but computationally infeasible) learning algorithm can exploit to perform much better than the upper bound given in Theorem 1. The proof of Theorem 3, which is contained in the supplement, constructs an example of such a family.

**Theorem 3.** *Suppose the loss function $L$ is 0-1 loss. There exists a label regularization function family $\mathcal{R}$ that satisfies Assumptions 1 and 2, but not Assumption 3, and a learning algorithm $A$ such that for all distributions $\mathcal{D}$ if $(\hat{\mathbf{x}}, \hat{\mathbf{y}}) \sim \mathcal{D}^m$ then with probability at least $1 - \delta$*

$$E_{\mathcal{D}}[L(\hat{\boldsymbol{\theta}}, x, y)] \leq \max_{\mathbf{q} \in \Delta^m} \left( E_{\hat{\mathbf{x}}, \mathbf{q}}[L(\hat{\boldsymbol{\theta}}, x, y)] - R(\mathbf{q}) \right) + \min_{\mathbf{q} \in \Delta^m} R(\mathbf{q}) + \epsilon(\delta, m) - 1$$

*for some $R \in \mathcal{R}(\hat{\mathbf{x}}, \hat{\mathbf{y}})$, where $\hat{\boldsymbol{\theta}}$ is the parameter output by $A$.*

Whenever $\lim_{m \to \infty} \epsilon(\delta, m) = 0$ the gap between the upper and lower bounds in Theorems 1 and 2 approaches $R(\hat{\mathbf{y}}) - \min_{\mathbf{q}} R(\mathbf{q})$ as $m \to \infty$ (ignoring constant factors). Therefore, these bounds are asymptotically matching if the labeler always chooses a label regularization function $R$ such that $R(\hat{\mathbf{y}}) = \min_{\mathbf{q}} R(\mathbf{q})$. We emphasize that this is true even if $\hat{\mathbf{y}}$ is a *nonunique* minimum of $R$. Several of the example learning settings described in Section 2.1, such as semi-supervised learning and ambiguous learning, meet this criteria. On the other hand, if $R(\hat{\mathbf{y}}) - \min_{\mathbf{q}} R(\mathbf{q})$ is large, then the gap is very large, and the utility of our analysis degrades. In the extreme case that $R(\hat{\mathbf{y}}) = \infty$ (i.e., the correct labeling of the training set is not possible under $R$), our upper bound is vacuous. In this sense, our framework is best suited to settings in which the information provided by the labeler is *equivocal*, but not actually *untruthful*, as it is in the malicious label noise setting [6, 7].

Finally, note that if $\lim_{m \to \infty} \epsilon(\delta, m) = 0$, then the upper bound in Theorem 3 is smaller than the lower bound in Theorem 2 for all sufficiently large $m$, which establishes the importance of Assumption 3.

## 4 Algorithm

Given the unlabeled training examples $\hat{\mathbf{x}}$ and label regularization function $R$, the bounds in Section 3 suggest an obvious learning algorithm: Find a parameter $\boldsymbol{\theta}^*$ that realizes the minimum

$$\min_{\boldsymbol{\theta}} \max_{\mathbf{q} \in \Delta^m} \left( E_{\hat{\mathbf{x}}, \mathbf{q}}[L(\boldsymbol{\theta}, x, y)] - R(\mathbf{q}) \right) + \alpha \left\| \boldsymbol{\theta} \right\|^2. \tag{1}$$

The objective (1) is simply the minimization of the upper bound in Theorem 1, with one difference: for algorithmic convenience, we do not minimize over the set $\Theta$, but instead add the quantity $\alpha \left\| \boldsymbol{\theta} \right\|^2$ to the objective and leave $\boldsymbol{\theta}$ unconstrained (here, and in the rest of the paper, $\left\| \cdot \right\|$ denotes $L_2$ norm). If we assume that $\Theta = \{ \boldsymbol{\theta} : \left\| \boldsymbol{\theta} \right\| \leq c \}$ for some $c > 0$, then this modification is without loss of generality, since there exists a constant $\alpha_c$ for which this is an equivalent formulation.

In order to estimate $\boldsymbol{\theta}^*$, throughout this section we make the following assumption about the loss function $L$ and label regularization function $R$.

**Assumption 4.** *The loss function $L$ is convex in $\boldsymbol{\theta}$, and the label regularization function $R$ is convex in $\mathbf{q}$.*

It is easy to verify that all of the loss functions and label regularization functions we gave as examples in Sections 2 and 2.1 satisfy Assumption 4.

Instead of finding $\boldsymbol{\theta}^*$ directly, our approach will be to "swap" the min and max in (1), find the soft labeling $\mathbf{q}^*$ that realizes the maximum, and then use $\mathbf{q}^*$ to compute $\boldsymbol{\theta}^*$. For convenience, we abbreviate the function that appears in the objective (1) as $F(\boldsymbol{\theta}, \mathbf{q}) \triangleq E_{\hat{\mathbf{x}}, \mathbf{q}}[L(\boldsymbol{\theta}, x, y)] - R(\mathbf{q}) + \alpha \left\| \boldsymbol{\theta} \right\|^2$. A high-level version of our learning algorithm — called GAME due to the use of a game-theoretic minimax theorem in its proof of correctness — is given in Algorithm 1; the implementation details for each step are given below Theorem 4.

---
**Algorithm 1** GAME: Game for Adversarially Missing Evidence
---
1: **Given:** Constants $\epsilon_1, \epsilon_2 > 0$.
2: Find $\tilde{\mathbf{q}}$ such that $\min_{\boldsymbol{\theta}} F(\boldsymbol{\theta}, \tilde{\mathbf{q}}) \geq \max_{\mathbf{q} \in \Delta^m} \min_{\boldsymbol{\theta}} F(\boldsymbol{\theta}, \mathbf{q}) - \epsilon_1$
3: Find $\tilde{\boldsymbol{\theta}}$ such that $F(\tilde{\boldsymbol{\theta}}, \tilde{\mathbf{q}}) \leq \min_{\boldsymbol{\theta}} F(\boldsymbol{\theta}, \tilde{\mathbf{q}}) + \epsilon_2$
4: **Return:** Parameter estimate $\tilde{\boldsymbol{\theta}}$.

---

In the first step of Algorithm 1, we modify the objective (1) by swapping the min and max, and then find a soft labeling $\tilde{\mathbf{q}}$ that approximately maximizes this modified objective. In the next step, we

find a parameter $\tilde{\boldsymbol{\theta}}$ that approximately minimizes the original objective with respect to the fixed soft labeling $\tilde{\mathbf{q}}$. The next theorem proves that Algorithm 1 produces a good estimate of $\boldsymbol{\theta}^*$, the minimum of the objective (1). Its proof is in the supplement.

**Theorem 4.** *The parameter $\tilde{\boldsymbol{\theta}}$ output by Algorithm 1 satisfies* $\|\tilde{\boldsymbol{\theta}} - \boldsymbol{\theta}^*\| \leq \sqrt{\frac{8}{\alpha}(\epsilon_1 + \epsilon_2)}$.

We now briefly explain how the steps of Algorithm 1 can be implemented using off-the-shelf algorithms. For concreteness, we focus on an implementation for the loss function $L = L_{\text{like}}$, which is also the loss function we use in our experiments in Section 5.

The second step of Algorithm 1 is the easier one, so we explain it first. In this step, we need to minimize $F(\boldsymbol{\theta}, \tilde{\mathbf{q}})$ over $\boldsymbol{\theta}$. Since $\tilde{\mathbf{q}}$ is fixed in this minimization, we can ignore the $R(\tilde{\mathbf{q}})$ term in the definition of $F$, and we see that this minimization amounts to maximizing the likelihood of a log-linear model. This is a very well-studied problem, and there are numerous efficient methods available for solving it, such as stochastic gradient descent.

The first step of Algorithm 1 is more complicated, as it requires finding the maximum of a maxmin objective. Our approach is to first take the dual of the inner minimization; after doing this the function to maximize becomes $G(\mathbf{p}, \mathbf{q}) \triangleq H(\mathbf{p}) - \frac{1}{\alpha} \|\Delta_{\boldsymbol{\phi}}(\mathbf{p}, \mathbf{q})\|^2 - R(\mathbf{q})$, where we let $H(\mathbf{p}) \triangleq -\sum_{i,y} p(i,y) \log p(i,y)$ and $\Delta_{\boldsymbol{\phi}}(\mathbf{p}, \mathbf{q}) \triangleq E_{\hat{\mathbf{x}}, \mathbf{p}}[\boldsymbol{\phi}(x,y)] - E_{\hat{\mathbf{x}}, \mathbf{q}}[\boldsymbol{\phi}(x,y)]$. By convex duality we have $\max_{\mathbf{q}} \min_{\boldsymbol{\theta}} F(\boldsymbol{\theta}, \mathbf{q}) = \max_{\mathbf{p}, \mathbf{q}} G(\mathbf{p}, \mathbf{q})$. This dual has been previously derived by several authors; see [15] for more details. Note that $G$ is concave function, and we need to maximize it over simplex constraints. Exponentiated-gradient-style algorithms [16, 15] are well-suited for this kind of problem, as they "natively" maintain the simplex constraint, and converged quickly in the experiments described in Section 5.

# 5 Experiments

We tested our GAME algorithm (Algorithm 1) on several standard learning data sets. In all of our experiments, we labeled a fraction of the training examples sets in a non-random manner that was designed to simulate various types of difficult — even adversarial — labelers.

Our first set of experiments involved two binary classification data sets that belong to a benchmark suite[1] accompanying a widely-used semi-supervised learning book [1]: the Columbia object image library (COIL) [17], and a data set of EEG scans of a human subject connected to a brain-computer interface (BCI) [18]. For each data set, a training set was formed by randomly sampling a subset of the data in a way that produced a skewed class distribution. We defined the *outlier score* of a training example to be the fraction of its nearest neighbors that belong to a different class. For several values of $p \in [0, 1]$ and for each training set, we labeled only the $p$-fraction of examples with the *highest* outlier score. In this way, we simulated an "unhelpful" labeler who only labels examples that are exceptions to the general rule, thinking (perhaps sincerely, but erroneously) that this is the most effective use of her effort.

We tested three algorithms on these data sets: GAME, where $\mathcal{R}(\hat{\mathbf{x}}, \hat{\mathbf{y}})$ was chosen to match the semi-supervised learning setting with a Laplacian regularizer (see Section 2.1); Laplacian SVM [3]; and Transductive SVM [19]. When constructing the Laplacian matrix and choosing values for hyperparameters, we adhered closely to the model-selection procedure described in [1, Sections 21.2.1 and 21.2.5]. The results of our experiments are given in Figures 1(a) and 1(b).

We also tested the GAME algorithm on a multiclass data set, namely a subset of the Labeled Faces in the Wild data set [20], a standard corpus of face photographs. Our subset contained 500 faces of the top 10 characters from the corpus, but with a randomly skewed distribution, so that some faces appeared more often than others. The feature representation for each photograph was PCA on the pixel values (i.e., eigenfaces). We used an ambiguously-labeled version of this data set, where each face in the training set is associated with one or more labels, only one of which is correct (see Section 2.1 for a definition of ambiguous learning). We labeled trainined examples to simulate a "lazy" labeler, in the following way: For each pair of labels $(y, y')$, we sorted the examples with true

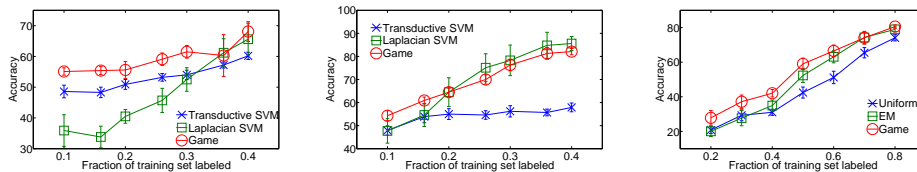

Figure 1: (a) Accuracy vs. fraction of unlabeled data for BCI data set. (b) Accuracy vs. fraction of unlabeled data for COIL data set. (c) Accuracy vs. fraction of partially labeled data for Faces in the Wild data set. In all plots, error bars represent 1 standard deviation over 10 trials.

label $y$ with respect to their distance, in feature space, from the centroid of the cluster of examples with true label $y'$. For several values of $p \in [0, 1]$, we added the label $y'$ to the top $p$-fraction of this list. The net effect of this procedure is that examples on the "border" of the two clusters are given *both* labels $y$ and $y'$ in the training set. The idea behind this labeling procedure is to mimic a (realistic, in our view) situation where a "lazy" labeler declines to commit to one label for those examples that are especially difficult to distinguish.

We tested the GAME algorithm on this data set, where $\mathcal{R}(\hat{\mathbf{x}}, \hat{\mathbf{y}})$ was chosen to match the ambiguous learning setting with a Laplacian regularizer (see Section 2.1). We compared with two algorithms from [9]: UNIFORM, which assumes each label in the ambiguous label set is equally likely, and learns a maximum likelihood log-linear model; and a discrimitive EM algorithm that guesses the true labels, learns the most likely parameter, updates the guess, and repeats. The results of our experiments are given in Figure 1(c).

Perhaps the best way to characterize the difference between GAME and the algorithms we compared it to is that the other algorithms are "optimistic", by which we mean they assume that the missing labels most likely agree with the estimated parameter, while GAME is a "pessimistic" algorithm that, because it was designed for an adverarial setting, assumes exactly the opposite. The results of our experiments indicate that, for certain labeling styles, as the fraction of fully labeled examples decreases, the GAME algorithm's pessimistic approach is substantially more effective. Importantly, Figures 1(a)-(c) show that the GAME algorithm's performance advantage is most significant when the number of labeled examples is very small. Semi-supervised learning algorithms are often promoted as being able to learn from only a handful of labeled examples. Our results show that this ability may be quite sensitive to how these examples are labeled.

# 6   Future Work

Our framework lends itself to several natural extensions. For example, it can be straightforwardly extended to the *structured prediction* setting [21], in which both examples and labels have some internal structure, such as sequences or trees. One can show that both steps of the GAME algorithm can be implemented efficiently even when the number of labels is combinatorial, provided that both the loss function and label regularization function decompose appropriately over the structure. Another possibility is to *interactively* poll the labeler for label information, resulting in a sequence of successively more informative label regularization functions, with the aim of extracting the most useful label information from the labeler with a minimum of labeling effort. Also, it would be interesting to design Amazon Mechanical Turk experiments that test whether the "unhelpful" and "lazy" labeling styles described in Section 5 in fact occur in practice. Finally, of the three technical assumptions we introduced in Section 3 to aid our analysis, we only proved (in Theorem 3) that one of them is necessary. We would like to determine whether the other assumptions are necessary as well, or can be relaxed.

### Acknowledgements

Umar Syed was partially supported by DARPA CSSG 2009 Award. Ben Taskar was partially supported by DARPA CSSG 2009 Award and the ONR 2010 Young Investigator Award.

## Footnotes

[1]This benchmark suite contains several data sets; we selected these two because they contain a large number of examples that meet our definition of outliers.

# References

[1] Olivier Chapelle, Bernhard Schölkopf, and Alexander Zien, editors. *Semi-Supervised Learning*. MIT Press, Cambridge, MA, 2006.

[2] Thomas G. Dietterich, Richard H. Lathrop, and Tomás Lozano-Pérez. Solving the multiple instance problem with axis-parallel rectangles. *Artificial Intelligence*, 89(1-2):31–71, 1997.

[3] Mikhail Belkin, Partha Niyogi, and Vikas Sindhwani. Manifold regularization: A geometric framework for learning from labeled and unlabeled examples. *Journal of Machine Learning Research*, 7:2399–2434, 2006.

[4] Gregory Druck, Gideon Mann, and Andrew McCallum. Learning from labeled features using generalized expectation criteria. In *Proceedings of the 31st Annual International ACM SIGIR Conference on Research and Development in Information Retrieval*, pages 595–602, 2008.

[5] Michael Kearns and Ming Li. Learning in the presence of malicious errors. In *Proceedings of the 20th Annual ACM Symposium on Theory of Computing*, pages 267–280, New York, NY, USA, 1988. ACM.

[6] Adam T. Kalai, Adam R. Klivans, Yishay Mansour, and Rocco A. Servedio. Agnostically learning halfspaces. In *Proceedings of the 46th Annual IEEE Symposium on Foundations of Computer Science*, pages 11–20, 2005.

[7] Adam R. Klivans, Philip M. Long, and Rocco A. Servedio. Learning halfspaces with malicious noise. *Journal of Machine Learning Research*, 10:2715–2740, 2009.

[8] Maria-Florina Balcan and Avrim Blum. A PAC-style model for learning from labeled and unlabeled data. In *Proceedings of the 18th Annual Conference on Learning Theory*, pages 111–126, 2005.

[9] Rong Jin and Zoubin Ghahramani. Learning with multiple labels. In *Advances in Neural Information Processing Systems 16*, 2003.

[10] Timothee Cour, Ben Sapp, Chris Jordan, and Ben Taskar. Learning from ambiguously labeled images. In *IEEE Computer Society Conference on Computer Vision and Pattern Recognition*, 2009.

[11] Kuzman Ganchev, João Graça, Jennifer Gillenwater, and Ben Taskar. Posterior regularization for structured latent variable models. *Journal of Machine Learning Research*, 11:2001–2049, 2010.

[12] Percy Liang, Michael I. Jordan, and Dan Klein. Learning from measurements in exponential families. In *Proceedings of the 26th Annual International Conference on Machine Learning*, pages 641–648, 2009.

[13] Rie Johnson and Tong Zhang. On the effectiveness of laplacian normalization for graph semi-supervised learning. *Journal of Machine Learning Research*, 8:1489–1517, December 2007.

[14] Philippe Rigollet. Generalization error bounds in semi-supervised classification under the cluster assumption. *Journal of Machine Learning Research*, 8:1369–1392, December 2007.

[15] Michael Collins, Amir Globerson, Terry Koo, Xavier Carreras, and Peter L. Bartlett. Exponentiated gradient algorithms for conditional random fields and max-margin markov networks. *Journal of Machine Learning Research*, 9:1775–1822, 2008.

[16] Jyrki Kivinen and Manfred K. Warmuth. Exponentiated gradient versus gradient descent for linear predictors. *Inf. Comput.*, 132(1):1–63, 1997.

[17] Sameer A. Nene, Shree K. Nayar, and Hiroshi Murase. Columbia object image library (COIL-100). Technical Report CUCS-006-96, Columbia University, 1996.

[18] Thomas Navin Lal, Thilo Hinterberger, Guido Widman, Michael Schröder, N. Jeremy Hill, Wolfgang Rosenstiel, Christian Erich Elger, Bernhard Schölkopf, and Niels Birbaumer. Methods towards invasive human brain computer interfaces. In *Advances in Neural Information Processing Systems 17*, 2004.

[19] Thorsten Joachims. Transductive inference for text classification using support vector machines. In *Proceedings of the 16th International Conference on Machine Learning*, pages 200–209, 1999.

[20] Gary B. Huang, Manu Ramesh, Tamara Berg, and Erik Learned-Miller. Labeled faces in the wild: A database for studying face recognition in unconstrained environments.

[21] Ben Taskar, Carlos Guestrin, and Daphne Koller. Max-margin markov networks. In *Advances in Neural Information Processing Systems 16*, 2004.

